# Learning Cellular Automaton Dynamics with Neural Networks

N H Wulff[*] and J A Hertz[†]
CONNECT, the Niels Bohr Institute and Nordita
Blegdamsvej 17, DK-2100 Copenhagen Ø, Denmark

## Abstract

We have trained networks of $\Sigma - \Pi$ units with short-range connections to simulate simple cellular automata that exhibit complex or chaotic behaviour. Three levels of learning are possible (in decreasing order of difficulty): learning the underlying automaton rule, learning asymptotic dynamical behaviour, and learning to extrapolate the training history. The levels of learning achieved with and without weight sharing for different automata provide new insight into their dynamics.

## 1  INTRODUCTION

Neural networks have been shown to be capable of learning the dynamical behaviour exhibited by chaotic time series composed of measurements of a single variable among many in a complex system [1, 2, 3]. In this work we consider instead cellular automaton arrays (CA)[4], a class of many-degree-of-freedom systems which exhibits very complex dynamics, including universal computation. We would like to know whether neural nets can be taught to imitate these dynamics, both locally and globally.

One could say we are turning the usual paradigm for studying such systems on its head. Conventionally, one is given the rule by which each automaton updates its state, and the (nontrivial) problem is to find what kind of global dynamical

---

[*]Present address: NEuroTech A/S, Copenhagen, Denmark

[†]Address until October 1993: Laboratory of Neuropsychology, NIMH, Bethesda MD 20892. email: hertz@nordita.dk

behaviour results. Here we suppose that we are given the history of some CA, and we would like, if possible, to find the rule that generated it.

We will see that a network can have different degrees of success in this task, depending on the constraints we place on the learning. Furthermore, we will be able to learn something about the dynamics of the automata themselves from knowing what level of learning is possible under what constraints.

This note reports some preliminary investigations of these questions. We study only the simplest automata that produce chaotic or complex dynamic behaviour. Nevertheless, we obtain some nontrivial results which lead to interesting conjectures for future investigation.

A CA is a lattice of formal computing units, each of which is characterized by a state variable $S_i(t)$, where $i$ labels the site in the lattice and $t$ is the (digital) time. Every such unit updates itself according to a particular rule or function $f(\ )$ of its own state and that of the other units in its local neighbourhood. The rule is the same for all units, and the updatings of all units are simultaneous.

Different models are characterized by the nature of the state variable (e.g. binary, continuous, vector, etc), the dimensionality of the lattice, and the choice of neighbourhood. In the two cases we study here, the neighbourhoods are of size $N = 3$, consisting of the unit itself and its two immediate neighbours on a chain, and $N = 9$, consisting of the unit itself and its 8 nearest neighbours on a square lattice (the 'Moore neighbourhood'). We will consider only binary units, for which we take $S_i(t) = \pm 1$. Thus, if the neighbourhood (including the unit itself) includes $N$ sites, $f(\ )$ is a Boolean function on the $N$-hypercube. There are $2^{2^N}$ such functions.

Wolfram [4] has divided the rules for such automata further into three classes:

1. Class 1: rules that lead to a uniform state.
2. Class 2: rules that lead to simple stable or periodic patterns.
3. Class 3: rules that lead to chaotic patterns.
4. Class 4: rules that lead to complex, long-lived transient patterns.

Rules in the fourth class lie near (in a sense not yet fully understood [5]) a critical boundary between classes 2 and 3. They lead eventually to asymptotic behaviour in class 2 (or possibly 3); what distinguishes them is the length of the transient. It is classes 3 and 4 that we are interested in here.

More specifically, for class 3 we expect that after the (short) initial transients, the motion is confined to some sort of attractor. Different attractors may be reached for a given rule, depending on initial conditions. For such systems we will focus on the dynamics on these attractors, not on the short transients. We will want to know what we can learn from a given history about the attractor characterizing it, about the asymptotic dynamics of the system generally (i.e. about all attractors), and, if possible, about the underlying rule.

For class 4 CA, in contrast, only the transients are of interest. Different initial conditions will give rise to very different transient histories; indeed, this sensitivity is the dynamical basis for the capability for universal computation that has been

proved for some of these systems. Here we will want to know what we can learn from a portion of such a history about its future, as well as about the underlying rule.

## 2   REPRESENTING A CA AS A NETWORK

Any Boolean function of $N$ arguments can be implemented by a $\Sigma-\Pi$ unit of order $P \leq N$ with a threshold activation function, i.e. there exist weights $w^0_{j_1 j_2 \cdots j_P}$ such that

$$f(S_1, S_2 \cdots S_N) = \text{sgn} \left[ \sum_{j_1, j_2, \cdots j_P} w^0_{j_1 j_2 \cdots j_P} S_{j_1} S_{j_2} \cdots S_{j_P} \right]. \qquad (1)$$

The indices $j_k$ run over the sites in the neighbourhood (1 to $N$) and zero, which labels a constant formal bias unit $S_0 = 1$. Because the updating rule we are looking for is the same for the entire lattice, the weight $w^0_{j_1 \cdots j_P}$ doesn't depend on $i$. Furthermore, because of the discrete nature of the outputs, the weights that implement a given rule are not unique; rather, there is a region of weight space for each rule.

Although we could work with other architectures, it is natural to study networks with the same structure as the CA to be simulated. We therefore make a lattice of formal $\Sigma-\Pi$ neurons with short-range connections, which update themselves according to

$$V_i(t+1) = g \left[ \sum_{j_1 \cdots j_P} w_{j_1 \cdots j_P} V_{j_1}(t) \cdots V_{j_P}(t) \right], \qquad (2)$$

In these investigations, we have assumed that we know *a priori* what the relevant neighbourhood size is, thereby fixing the connectivity of the network. At the end of the day, we will take the limit where the gain of the activation function $g$ becomes infinite. However, during learning we use finite gain and continuous-valued units.

We know that the order $P$ of our $\Sigma-\Pi$ units need not be higher than the neighbourhood size $N$. However, in most cases a smaller $P$ will do. More precisely, a network with any $P > \frac{1}{2}N$ can in principle (i.e. given the right learning algorithm and sufficient training examples) implement almost all possible rules. This is an asymptotic result for large $N$ but is already quite accurate for $N = 3$, where only two of the 256 possible rules are not implementable by a second-order unit, and $N = 5$, where we found from simple learning experiments that 99.87% of 10000 randomly-chosen rules could be implemented by a third-order unit.

## 3   LEARNING

Having chosen a suitable value of $P$, we can begin our main task: training the network to simulate a CA, with the training examples $\{S_i(t) \rightarrow S_i(t+1)\}$ taken from a particular known history.

The translational invariance of the CA suggests that weight sharing is appropriate in the learning algorithm. On the other hand, we can imagine situations in which we did not possess *a priori* knowledge that the CA rule was the same for all units,

or where we only had access to the automaton state in one neighbourhood. This case is analogous to the conventional time series extrapolation paradigm, where we typically only have access to a few variables in a large system. The difference is that here the accessible variables are binary rather than continuous. In these situations we should or are constrained to learn without each unit having access to error information at other units. In what follows we will perform the training both with and without weight sharing. The differences in what can be learned in the two cases will give interesting information about the CA dynamics being simulated.

Most of our results are for chaotic (class 3) CA. For these systems, this training history is taken after initial transients have died out. Thus many of the $2^N$ possible examples necessary to specify the rule at each site may be missing from the training set, and it is possible that our training procedure will not result in the network learning the underlying rule of the original system. It might instead learn another rule that coincides with the true one on the training examples. This is even more likely if we are not using weight sharing, because then a unit at one site does not have access to examples from the training history at other sites.

However, we may relax our demand on the network, asking only that it evolve exactly like the original system when it is started in a configuration the original system could be in after transients have died out (i.e. on an attractor of the original system). Thus we are restricting the test set in a way that is "fairer" to the network, given the instruction it has received.

Of course, if the CA has more than one attractor, several rules which yield the same evolution on one attractor need not do so on another one. It is therefore possible that a network can learn the attractor of the training history (i.e. will simulate the original system correctly on a part of the history subsequent to the training sequence) but will not be found to evolve correctly when tested on data from another attractor.

For class 4 automata, we cannot formulate the distinctions between different levels of learning meaningfully in terms of attractors, since the object of interest is the transient portion of the history. Nevertheless, we can still ask whether a network trained on part of the transient can learn the full rule, whether it can simulate the dynamics for other initial conditions, or whether it can extrapolate the training history.

We therefore distinguish three degrees of successful learning:

1. *Learning the rule*, where the network evolves exactly like the original system *from any initial configuration*.

2. *Learning the dynamics*, the intermediate case where the network can simulate the original system exactly after transients, irrespective of initial conditions, despite not having learned the full rule.

3. *Learning to continue the dynamics*, where the successful simulation of the original system is only achieved for the particular initial condition used to generate the training history.

Our networks are recurrent, but because they have no hidden units, they can be trained by a simple variant of the delta-rule algorithm. It can be obtained formally

from gradient descent on a modified cross entropy

$$E = \frac{1}{2} \sum_{it} \left[ (1 + S_i(t)) \log \frac{1 + S_i(t)}{1 + V_i(t)} + (1 - S_i(t)) \log \frac{1 - S_i(t)}{1 - V_i(t)} \right] \Theta[-S_i(t)V_i(t)] \quad (3)$$

We used the online version:

$$\Delta w_{j_1 j_2 \cdots j_P} = \eta \Theta[-S_i(t+1)V_i(t+1)][S_i(t+1) - V_i(t+1)]V_{j_1}(t)V_{j_2}(t) \cdots V_{j_P}(t) \quad (4)$$

This is like an extension of the *Adatron* algorithm[6] to $\Sigma - \Pi$ units, but with the added feature that we are using a nonlinear activation function.

The one-dimensional $N = 3$ automata we simulated were the 9 *legal* chaotic ones identified by Wolfram [4]. Using his system for labeling the rules, these are rules 18, 22, 54, 90, 122, 126, 146, 150, and 182. We used networks of order $P = 3$ so that all rules were learnable. (Rule 150 would not have been learnable by a second-order net.) Each network was a chain 60 units long, subjected to periodic boundary conditions.

The training histories $\{S_i(t)\}$ were 1000 steps long, beginning 100 steps after randomly chosen initial configurations. To test for learning the rules, all neighbourhood configurations were checked at every site. To test for learning the dynamics, the CA were reinitialized with different random starting configurations and run 100 steps to eliminate transients, after which new test histories of length 100 steps were constructed. Networks were then tested on 100 such histories. The test set for continuing the dynamics was made simply by allowing the CA that had generated the training set to continue for 100 more steps.

There are no class 4 CA among the one-dimensional $N = 3$ systems. As an example of such a rule, we chose the *Game of Life* which is defined on a square lattice with a neighbourhood size $N = 9$ and has been proved capable of universal computation (see, e.g. [7, 8]). We worked with a lattice of $60 \times 60$ units.

The training history for the Game of Life consisted of 200 steps in the transient. The trained networks were tested, as in the case of the chaotic one-dimensional systems, on all possible configurations at every site (learning the rule), on other transient histories generated from different initial conditions (learning the dynamics), and on the evolution of the original system immediately following the training history (learning to continue the dynamics).

## 4    RESULTS

With weight sharing, it proved possible to learn the dynamics for all 9 of the one-dimensional chaotic rules very easily. In fact, it took no more than 10 steps of the training history to achieve this.

Learning the underlying rules proved harder. After training on the histories of 1000 steps, the networks were able to do so in only 4 of the 9 cases. No qualitative difference in the two groups of patterns is evident to us from looking at their histories (Fig. 1). Nevertheless, we conclude that their ergodic properties must be different, at least quantitatively.

Life was also easy with weight sharing. Our network succeed in learning the underlying rule starting almost anywhere in the long transient.

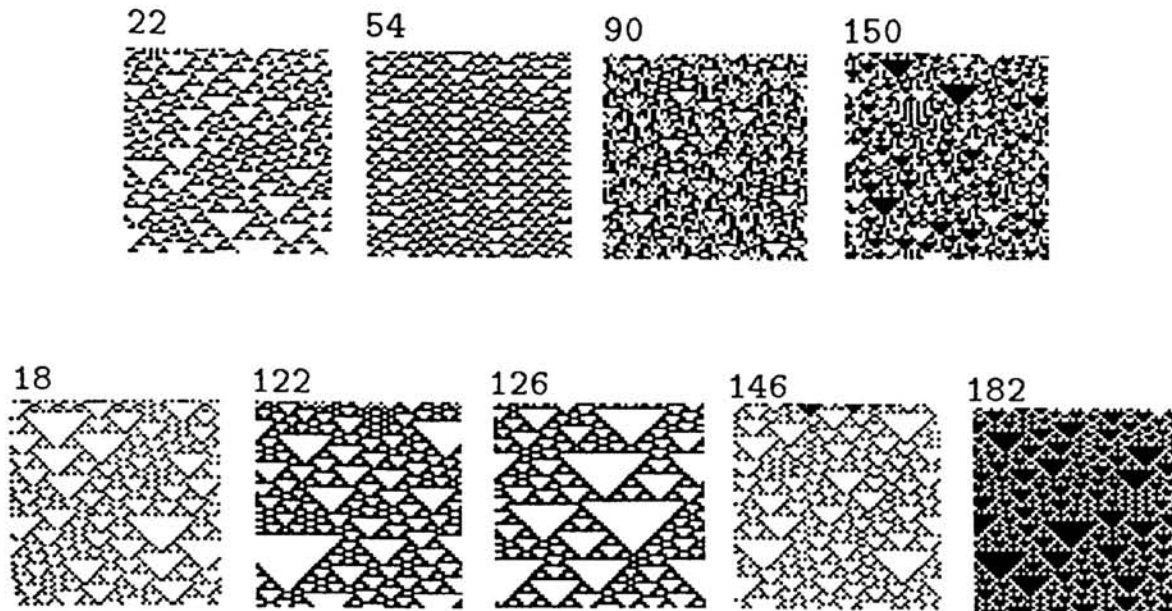

Figure 1: Histories of the 4 one-dimensional rules that could be learned (top) and the 5 that could not (bottom). (Learning with weight sharing.)

Without weight sharing, all learning naturally proved more difficult. While it was possible to learn to continue the dynamics for all the one-dimensional chaotic rules, it proved impossible except in one case (rule 22) to learn the dynamics within the training history of 1000 steps. The networks failed on about 25% of the test histories. It was never possible to learn the underlying rule. Thus, apparently these chaotic states are not as homogeneous as they appear (at least on the time scale of the training period).

Life is also difficult without weight sharing. Our network was unable even to continue the dynamics from histories of several hundred steps in the transient (Fig. 2).

## 5   DISCUSSION

In previous studies of learning chaotic behaviour in single-variable time series (e.g. [1, 2, 3]), the test to which networks have been put has been to extrapolate the training series, i.e. to continue the dynamics. We have found that this is also possible in cellular automata for all the chaotic rules we have studied, even when only local information about the training history is available to the units. Thus, the CA evolution history *at any site* is rich enough to permit error-free extrapolation.

However, local training data are not sufficient (except in one system, rule 22) to permit our networks to pass the more stringent test of learning the dynamics. Thus, viewed from any single site, the different attractors of these systems are dissimilar enough that data from one do not permit generalization to another.

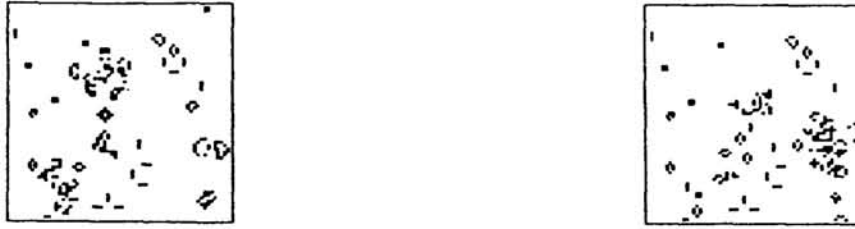

Figure 2: The original Game of Life CA (left) and the network (right), both 20 steps after the end of the training history. (Training done without weight sharing.)

With the access to training data from other sites implied by weight sharing, the situation changes dramatically. Learning the dynamics is then very easy, implying that all possible asymptotic local dynamics that could occur for any initial condition actually do occur *somewhere in the system* in any given history.

Furthermore, with weight sharing, not only the dynamics but also the underlying rule can be learned for some rules. This suggests that these rules are ergodic, in the sense that all configurations occur somewhere in the system at some time. This division of the chaotic rules into two classes according to this global ergodicity is a new finding.

Turning to our class 4 example, Life proves to be impossible without weight sharing, even by our most lenient test, continuing the dynamics. Thus, although one might be tempted to think that the transient in Life is so long that it can be treated operationally as if it were a chaotic attractor, it cannot. For real chaotic attractors, in both in the CA studied here and continuous dynamical systems, networks can learn to continue the dynamics on the basis of local data, while in Life they cannot.

On the other hand, the result that the rule of Life is easy to learn with weight sharing implies that looked at globally, the history of the transient is quite rich. Somewhere in the system, it contains sufficient information (together with the a *priori* knowledge that a second-order network is sufficient) to allow us to predict the evolution from any configuration correctly.

This study is a very preliminary one and raises more questions than it answers. We would like to know whether the results we have obtained for these few simple systems are generic to complex and chaotic CA. To answer this question we will have to study systems in higher dimensions and with larger updating neighbourhoods. Perhaps significant universal patterns will only begin to emerge for large neighborhoods (cf [5]). However, we have identified some questions to ask about these problems.

# References

[1] A Lapedes and R Farber, *Nonlinear Signal Processing Using Neural Networks: Prediction and System Modelling*, Tech Rept LA-UR-87-2662, Los Alamos National Laboratory. Los Alamos NM USA

[2] A S Weigend, B A Huberman and D E Rumelhart, *Int J Neural Systems* **1** 193-209 (1990)

[3] K Stokbro, D K Umberger and J A Hertz, *Complex Systems* **4** 603-622 (1991)

[4] S Wolfram, *Theory and Applications of Cellular Automata* (World Scientific, 1986)

[5] C G Langton, pp 12-37 in *Emergent Computation* (S Forrest, ed) MIT Press/North Holland, 1991

[6] J K Anlauf and M Biehl, *Europhys Letters* **10** 687 (1989)

[7] H V McIntosh, *Physica D* **45** 105-121 (1990)

[8] S Wolfram, *Physica D* **10** 1-35 (1984)
